# Further Explorations in Visually-Guided Reaching: Making MURPHY Smarter

**Bartlett W. Mel**
*Center for Complex Systems Research*
*Beckman Institute, University of Illinois*
*405 North Matheus Street*
*Urbana, IL 61801*

## ABSTRACT

MURPHY is a vision-based kinematic controller and path planner based on a connectionist architecture, and implemented with a video camera and Rhino XR-series robot arm. Imitative of the layout of sensory and motor maps in cerebral cortex, MURPHY's internal representations consist of four coarse-coded populations of simple units representing both static and dynamic aspects of the sensory-motor environment. In previously reported work [4], MURPHY first learned a direct kinematic model of his camera-arm system during a period of extended practice, and then used this "mental model" to heuristically guide his hand to unobstructed visual targets. MURPHY has since been extended in two ways: First, he now learns the inverse differential-kinematics of his arm in addition to ordinary direct kinematics, which allows him to push his hand directly towards a visual target without the need for search. Secondly, he now deals with the much more difficult problem of reaching in the presence of obstacles.

## INTRODUCTION

Visual guidance of a multi-link arm through a cluttered workspace is known to be an extremely difficult computational problem. Classical approaches in the field of robotics have typically broken the problem into pieces of manageable size, including modules for direct and inverse kinematics and dynamics [7], along with a variety of highly complex algorithms for motion planning in the configuration space of a multi-link arm (e.g. [3]). Workers in the field of robotics have rarely (until recently) emphasized neural plausibility at the level of representation and algorithm, opting instead for explicit mathematical computations or complex, multi-stage algorithms using general-purpose data structures. More peculiarly, very little emphasis has been placed on full use of the visual channel for robot control, other than as a source of target shape or coordinates.

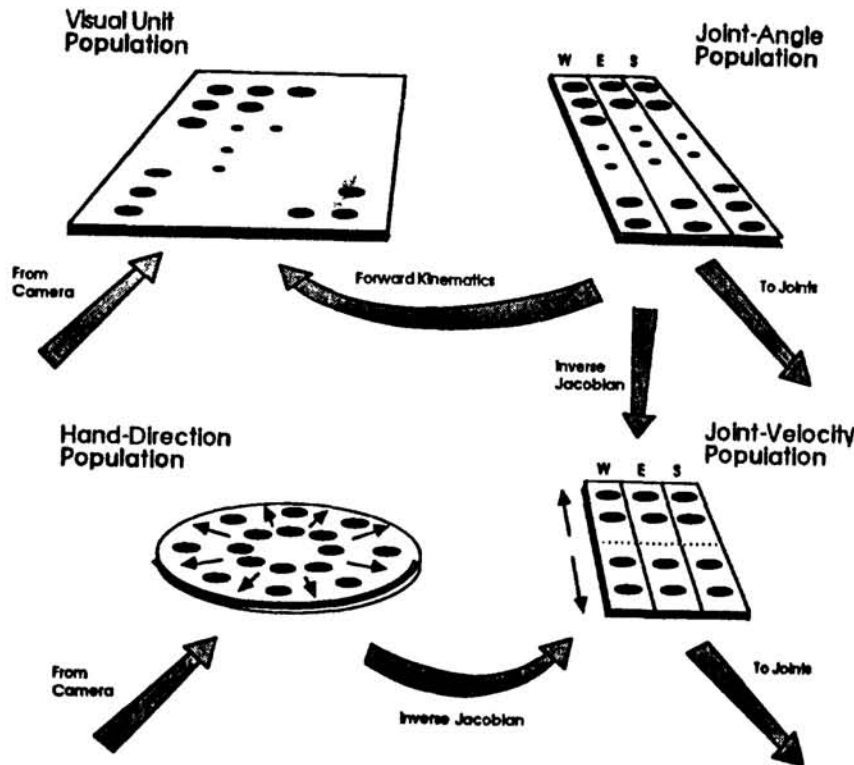

Figure 1: MURPHY's Connectionist Architecture. Four interconnected populations of neuron-like units implement a variety of sensory-motor mappings.

Much has been learned of the neural substrate for vision-guided limb control in humans and non-human primates (see [2] for review), albeit at a level too far removed from concrete algorithmic specification to be of direct engineering utility. Nonetheless, a number of general principles of cortical organization have inspired the current approach to vision-based kinematic learning and motion-planning. MUR-PHY's connectionist architecture has been based on the observation that a surprisingly large fraction of the vertebrate brain is devoted to the explicit representation of the animal's sensory and motor state [6]. During normal behavior, each of these neural representations carries behaviorally-relevant state information, some yoked to the sensory-epithelia, others to the motor system. The effect is a rich set of online associative learning opportunities. Moreover, the *visual* modality is by far the dominant in the primate brain by measures of sheer real-estate, including a number of areas that are known to be concerned with the representation of limb control in immediate extrapersonal space [2], suggesting that visual processing may overshadow what has usually been perceived as primarily a motor process.

## MURPHY's ORGANIZATION

In the interests of space, we present here a highly reduced description of MURPHY's organization; the reader is referred to [5] for a much more comprehensive treat-

ment, including a lengthy discussion of the relevance of MURPHY's structure and function to the psychology, motor-physiology, and neural-basis for visually-guided limb control in primates.

## The Physical Setup

MURPHY's physical setup consists of a 512 x 512 JVC video camera pointed at a Rhino XR-3 robotic arm, whose wrist, elbow, and shoulder rotate freely in the image plane of the camera. White spots are stuck to the arm in convenient places; when the image is thresholded, only the white spots appear in the image (see fig. 2). This arrangement allows continuous control over the complexity of the visual image of the arm, which in turn affects computation time during learning. The arm is software controllable, with a stepper motor for each joint. Arm dynamics are not dealt with in this work.

## The Connectionist Architecture

MURPHY is currently based on four interconnected populations of neuron-like units (fig. 1), encoding both static and dynamic aspects of the sensory-motor environment (only two were used in a previous report [4]). *Visual Populations*. The principal sensory population is organized as a rectangular, visuotopically-mapped 64 x 64 grid of coarsely-tuned visual units, each of which responds when a visual feature (such as a white spot on the arm) falls into its receptive field (fig 1, upper left). The second population of 24 units encodes the direction of MURPHY's hand motion through the visual field (fig. 1, lower left)—vector magnitude is ignored at present. These units are thus "fired" only by the distinct visual image of the hand, but are selective for the direction of hand motion through the visual field as MURPHY moves his arm in the workspace. *Joint Populations*. The third population of 273 units consists of three subpopulations encoding the static joint configuration; the angle of each joint is value-coded individually in a subpopulation dedicated to that joint, consisting of units with overlapping triangular receptive fields. (fig. 1, upper right). The fourth and final population of 24 units also consists of three subpopulations, each value-coding the *velocity* of one of the three joints (fig. 1, lower right).

During both his learning and performance phases to be described in subsequent sections, MURPHY is also able to carry out simple sequential operations that are driven by a control structure external to his connectionist architecture.

## MURPHY's Kinematics

For a detailed discussion of the relation between MURPHY's novel style of kinematic representation and those used in other approaches to robot control, see [5]. Briefly, in reference to the four unit populations described above, MURPHY's primary workhorse is his direct kinematic mapping that relates static joint angles to the associated visual image of the arm. This smooth nonlinear mapping comprises both the kinematics of the arm and the optical parameters and global geometry of

the camera/imaging system, and is learned and represented as a synaptic projection from the joint-angle to visual-field populations (fig. 1). Post-training, MURPHY can assume an arbitrary joint posture "mentally", i.e. by setting up the appropriate pattern of activation on his joint-angle population without allowing the arm to move. The learned mapping will then synaptically activate a mental image of the arm, in that configuration, on the "post-synaptic" visual-field population. Contemplated movements of the arm can thus be evaluated *without overt action*—this is the heart of MURPHY's mental model.

MURPHY is also able to learn the inverse differential-kinematics of his arm, a mapping which translates a desired direction of motion through the workspace into the requisite commands to the joints, allowing MURPHY to guide his hand along a desired trajectory through his field of view. This mapping is learned and represented as a synaptic projection originating from both i) the hand-vector population, encoding the desired visual-field direction, and ii) the joint-angle population encoding the current state of the arm, and terminating on the joint-move population, which specifies the appropriate pertubation to the joints (fig. 1, see arrows labelled "Inverse Jacobian"). In the next section, we describe how this learning takes place.

## HOW MURPHY LEARNS

As described in [4,5], MURPHY learns by doing. Thus, during an initial training period for the direct kinematics, MURPHY steps his arm systematically through a small representative sample of the 3.3 billion legal arm configurations (visiting 17,000 configs. in 5 hours). Each step constitutes a standard connectionist training example between his joint-angle and visual-field populations. A novel synaptic learning scheme called *sigma-pi learning* is used for weight modification [4,5], described elsewhere in great detail [5]. This scheme essentially treats each post-synaptic *sigma-pi* neuron (see [5]) as an interpolating lookup table of the kind discussed by Albus and others [1], rather than as a standard linear threshold unit. *Sigma-pi* learning has been inspired by the physical structure and membrane properties of biological neurons, and yields several advantages in performance and simplicity of implementation for the learning of smooth low-dimensional functions [5]. As an implementation note, once the *sigma-pi* units have been appropriately trained, they are reimplemented using *k-d trees*, a much more efficient data-structure for a sequential computer (giving a speedup on the order of 50-100).

MURPHY's inverse-differential mapping is learned analogously, where each *movement* of the arm (rather than each *state*) is used as a training example. Thus, as the hand sweeps through the visual field during either physical *or* mental practice, each of the three relevant populations are activated (hand-vector and joint-angle as inputs, joint-move as output), acting again as a single input-output training example for the learning procedure.

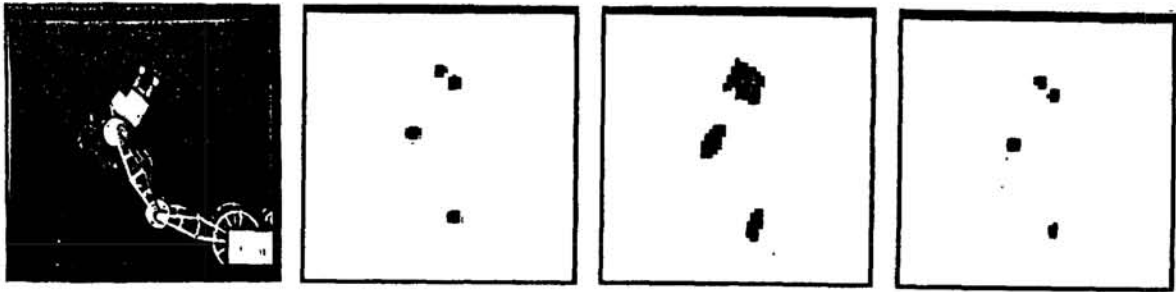

Figure 2: Four Visual Representations. The first frame shows the unprocessed camera view of MURPHY's arm. White spots have been stuck to the arm at various places, such that a thresholded image contains *only* the white spots. The second frame shows the resulting pattern of activation over the 64 x 64 grid of coarsely-tuned visual units as driven by the camera. The third frame depicts an internally-produced "mental" image of the arm in the same configuration, as driven by weighted connections from the joint-angle population. Note that the mental image is a sloppy, but highly recognizable approximation to the camera-driven trace. The fourth frame shows the mental image generated using k-d trees in the place of *sigma-pi* units.

# MURPHY IN ACTION

## Reaching to Targets

In a previous report, MURPHY was only able to reach to a visual target by mentally flailing his way to the target (i.e. by generating a small random change in joint position, evaluating the induced mental image of the arm for proximity to the target, and keeping only those moves that reduced this distance), and then moving the arm physically in one fell swoop [4]. On repeated reaches to the same or similar targets, MURPHY was doomed to repeatedly wander his way stupidly and aimlessly to the target. Typical trajectories generated in this way can be seen in fig. 3ABC. Using only the steps in these three trajectories as training examples for MURPHY's inverse-differential mapping, and then allowing this map to generate "guesses" as to the appropriate joint-move at each step, the trajectories for similar targets are substantially more direct (fig. 3DEF).

## Avoiding Obstacles

Augmenting this direct search approach with only a few additional visual heuristics, MURPHY is able to find circuitous paths through complicated obstacle layouts, even when contrived with significant local minima designed to trap the arm (fig. 4). For problems of this kind, MURPHY uses a non-replacement, best-first search with

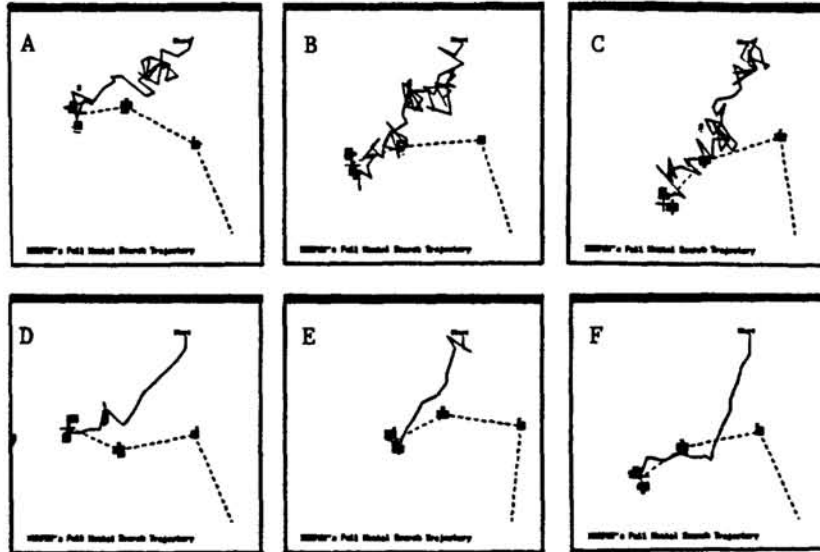

Figure 3: Improving with Practice. Frames A, B, and C shows MURPHY's initially random search trajectories from start to goal. Joint moves made during these three "mental" reaching episodes were used to train MURPHY's inverse differental-kinematic mapping. Frames D, E, and F show improvement in 3 subsequent reaching trials to nearby targets.

backtracking on a quantized grid in confuguration space. Mental images of the arm were generated in sequence, and evaluated according to several criteria: moves that brought the hand closer to the target without collision with obstacles were accepted, marked, and pursued; moves that either had been tried before, pushed the hand out of the visual field, or resulted in collision were rejected (i.e. popped). Collision detection, usually considered a combinatorially expensive operation under typical representational assumptions (see [3]), is here represented as a single, parallel virtual-machine operation that detects superposition between arbitrary obstacle-blobs in the visual field and the mental image of the arm. Problems such as that of fig. 4 consumed an average of 10 minutes on a Sun 3-160 running inefficiently with full graphics. Reaching trials only consistently failed when the grain of quantization in MURPHY's configuration space search prevented him from finding clear paths through too-tight spaces. This problem could be (but has not as yet been) attacked through hierarchical quantization.

## CONCLUSIONS

MURPHY's design has evolved from three schools of thought: ROBOTICS WITHOUT EQUATIONS, LEARNING WITHOUT TEACHERS, and BETTER LIVING THROUGH VISION. First, the approach illustrates that neurally-inspired representational structures can, without equations, implement the core functional-mappings used in robot control. The approach also demonstrates that a great deal of useful knowledge can be extracted from the environment without need of a teacher, i.e. simply by *do-*

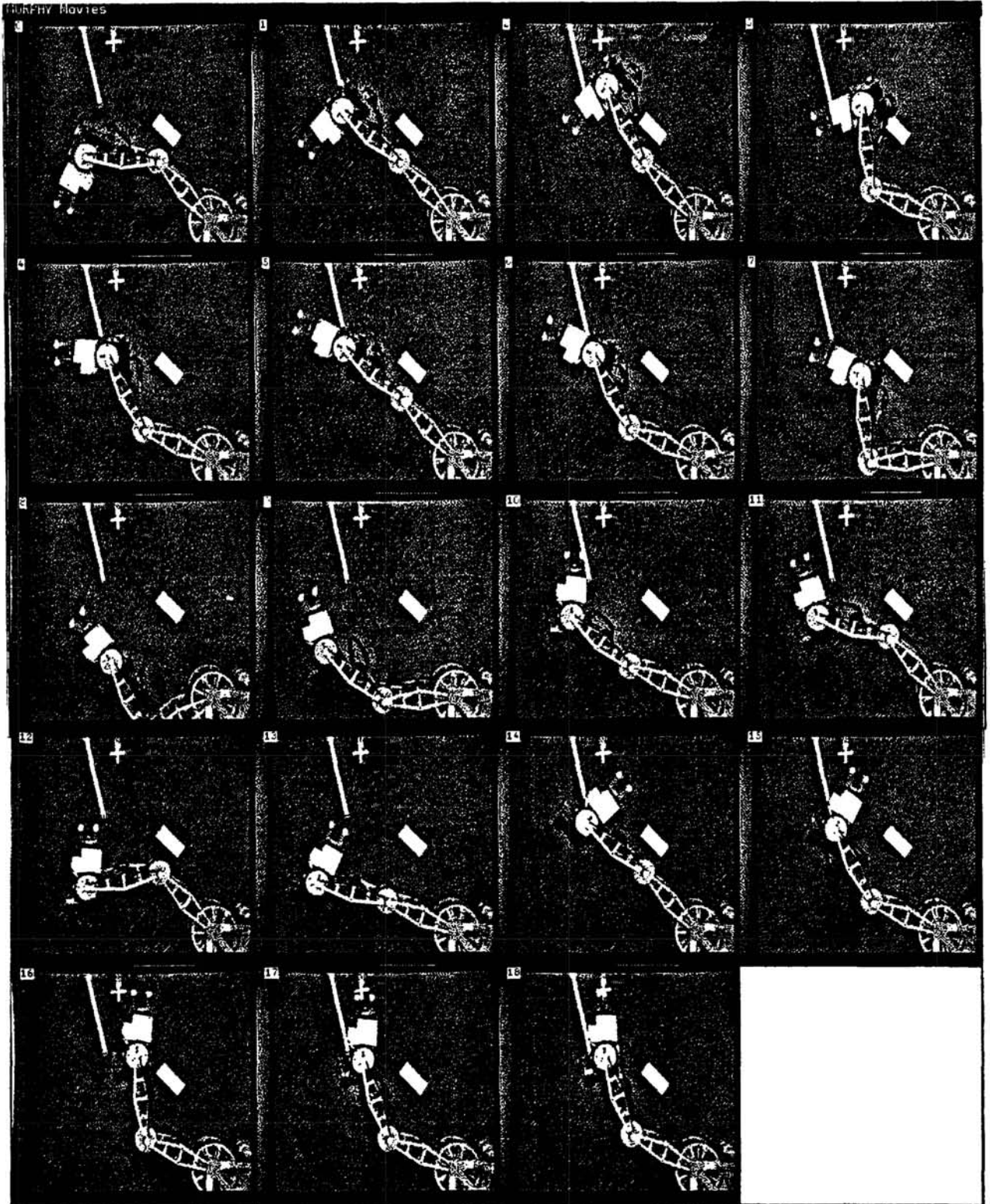

Figure 4: Reaching for a target (white cross) in the presence of obstacles (miscellaneous other white blobs). MURPHY typically used fewer than 100 internal search steps for problems of this approximate difficulty.

*ing.* Thirdly, the approach illustrates that planning can be naturally carried out simultaneously in joint *and* workspace coordinates, that is, can be "administered" in joint space, but evaluated using massively parallel visual machine operations. Thus, the use of a massively-parallel architecture makes direct heuristic search through the configuration space of an arm computationally feasible, since a single plan step (i.e. running the direct kinematics and evaluating for progress and/or collision) is reduced to $O(1)$ virtual machine operations. This feature of the approach is that which most distinguishes MURPHY from other motion-planning schemes.

A detailed analysis of the scaling behavior of this approach was carried out in [4] suggesting that a real-time, 3-d vision/6 degree-of-freedom super-MURPHY could be built with state-of-the-art 1988 hardware, though it must be stressed that the competitiveness of the approach depends heavily on massive hardware parallelism that is not conveniently available at this time. Questions also remain as to the scaling of *problem difficulty* in the jump to a practical real world systems.

## Acknowledgements

This work was supported in part by a University of Illinois Cognitive Science/AI fellowship, the National Center for Supercomputing Applications, Champaign, Illinois, and NSF grant Phy 86-58062. Thanks are also due to Stephen Omohundro for encouragement and scientific support throughout the course of the project.

# References

[1] Albus, J.S. A new approach to manipulator control: the cerebellar model articulation controller (CMAC). *ASME J. of Dynamic Systems, Measurement, & Control*, September 1975, 220-227.

[2] Humphrey, D.R. On the cortical control of visually directed reaching: contributions by nonprecentral motor areas. In *Posture and movement*, R.E. Talbott & D.R. Humphrey, (Eds.), New York: Raven Press, 1979.

[3] Lozano-Pérez, T. A simple motion-planning algorithm for general robot manipulators. *IEEE J. of Robotics & Automation*, 1987, RA-3, 224-238.

[4] Mel, B.W. MURPHY: A robot that learns by doing. In *Neural information processing systems*, p. 544-553, American Institute of Physics, New York, 1988.

[5] Mel, B.W. A neurally-inspired connectionist approach to learning and performance in vision-based robot motion planning. Technical Report CCSR-89-17, Center for Complex Systems Research, Beckman Institute, University of Illinois, 405 N. Matheus, Urbana, IL 61801.

[6] Merzenich, M.M & Kaas, J. Principles of organization of sensory-perceptual systems in mammals. In *Progress in psychobiology and physiological psychology*, vol. 9, 1980.

[7] Paul, R. *Robot manipulators: mathematics, programming, and control.* Cambridge: MIT Press, 1981.
